# Motif-oriented influence maximization for viral marketing in large-scale social networks

**Mingyang Zhou**
Guangdong Province Key Laboratory of
Popular High Performance Computers
College of Computer Science
and Software Engineering
Shenzhen University
Shenzhen, China 518060
zmy@szu.edu.cn

**Weiji Cao**
College of Computer Science
and Software Engineering
Shenzhen University
Shenzhen, China 518060
2020151062@email.szu.edu.cn

**Hao Liao**[*]
Guangdong Province Key Laboratory of
Popular High Performance Computers
College of Computer Science
and Software Engineering
Shenzhen University
Shenzhen, China 518060
haoliao@szu.edu.cn

**Rui Mao**
Guangdong Province Key Laboratory of
Popular High Performance Computers
College of Computer Science
and Software Engineering
Shenzhen University
Shenzhen, China 518060
mao@szu.edu.cn

## Abstract

The *influence maximization (IM)* problem aims to identify a budgeted set of nodes with the highest potential to influence the largest number of users in a cascade model, a key challenge in viral marketing. Traditional *IM* approaches consider each user/node independently as a potential target customer. However, in many scenarios, the target customers comprise motifs, where activating only one or a few users within a motif is insufficient for effective viral marketing, which, nevertheless, receives little attention. For instance, if a motif of three friends planning to dine together, targeting all three simultaneously is crucial for a restaurant advertisement to succeed. In this paper, we address the motif-oriented influence maximization problem under the linear threshold model. We prove that the motif-oriented IM problem is NP-hard and that the influence function is neither supermodular nor submodular, in contrast to the classical *IM* setting. To simplify the problem, we establish the submodular upper and lower bounds for the influence function. By leveraging the submodular property, we propose a natural greedy strategy that simultaneously maximizes both bounds. Our algorithm has an approximation ratio of $\tau \cdot (1 - 1/e - \varepsilon)$ and a near-linear time complexity of $O((k+l)(m+\eta) \log \eta / \varepsilon^2)$. Experimental results on diverse datasets confirm the effectiveness of our approach in motif maximization.

## 1 Introduction

The utilization of "word-of-mouth" and "viral marketing" techniques has become prevalent in the promotion of new products. In social networks, "viral marketing" is implemented by selecting highly

---

[*]Corresponding author

influential users as initial adopters [1, 2]. The ultimate goal of this strategy is to trigger widespread adoption of products along social connections, i.e., "*influence maximization (IM)*" problem [3]. *IM* serves as a key research problem in network analysis and has received significant attention due to its commercial applications [4, 5, 6]. While most studies perceive each user as a potential target consumer, our paper aims to explore a more general *IM* problem where the target consumers are represented as motifs, each motif consisting of multiple users [7, 8]. In this scenario, activating only one (or a few) users within a motif may prove ineffective for viral marketing.

Consider the following scenario: A restaurant seeks to attract customers through viral marketing on social networks. In this scenario, a motif of $k$ friends plans to dine together, and they have three different underlying mechanisms for choosing a restaurant: (a) One user already knows a restaurant and recommends it to others in the motif. Therefore, targeting only one user is sufficient for advertising the restaurant. (b) All users in the group must agree on the restaurant choice, requiring the advertisement to activate every member in the motif. (c) The decision is made through voting, with the restaurant chosen if more than half users in the motif agree. To maximize restaurant profits, advertisements should target entire motifs rather than individual users in social networks. Similar situations arise in other group decision-making scenarios, such as anonymous voting, family parties, and group tours [9, 10, 11]. When it comes to viral marketing aimed at motif consumers, maximizing the number of ultimate activated motifs is the explicit purpose for the *IM*, called motif-oriented IM.

The motif-oriented *IM* model differs from the classical *IM* in two key aspects: (a) A user can belong to multiple motifs and consume a product multiple times. (b) Activating the most users in the classical *IM* may only trigger a few motifs, while activating most motifs might require fewer users than that of the classical *IM*. These factors present challenges in designing an effective strategy to determine the seed nodes. Note that some works investigate the group-oriented *IM* [9], which is different from our motif-based *IM*. Motifs are the function units of a graph [7, 8] and users are mutually connected within a motif. Whereas users may be not connected (or partially reachable) in groups. In the paper, we provide an efficient solution for motif-based *IM* with a guaranteed approximation ratio.

This paper introduces an algorithm by optimizing the **l**ower and upper **b**ound of the **m**otif-**o**riented **i**nfluence **m**aximization (LBMOIM) to determine a budget set of seed nodes, aiming to maximize the number of activated motifs in social networks under the linear threshold model. In our model, each motif is associated with a threshold number $r_i$, and if the number of activated nodes in the motif exceeds $r_i$, the motif is called activated; otherwise, it remains inactive. Particularly, $r_i = \{1, k, k/2\}$ represents the three different restaurant determination scenarios introduced above. We define the target motif influence as the count of activated motifs. We find that the influence function is monotone, but is neither submodular nor supermodular, making direct optimization challenging. To address this, we establish the lower and upper bounds for the influence function for different $r_i$. The two bounds represent the weighted node-level influence function and both are monotone and submodular. We use a greedy algorithm to optimize the two bounds simultaneously to select the best seed nodes. Experiments in various datasets demonstrate the effectiveness and efficiency of our algorithm. In addition, our study demonstrates that LBMOIM can be expanded to accommodate various diffusion models [3].We introduce a modified version of LBMOIM for the independent cascade model. We provide evidence to confirm the applicability of LBMOIM in different cascade models.

## 2    Related work

*Node-level influence maximization.* Kempe et al. [3] first introduced an algorithmic study on the *IM* problem, demonstrating its NP-hardness in general cases. They proposed a greedy algorithm that can approximate an approximation ratio of $1 - 1/e$ with time complexity of $O(knmr)$. The major time-consuming step is the generation of Monte Carlo samples. To address this, various methods have been proposed to utilize *reverse reachable* (RR) sets rather than Monte Carlo simulation to reduce the time complexity without sacrificing the performance accuracy. These include the IMM [12], SSA [13], TIM [14], and OPTIM-C [15] techniques. More recently, Guo et al. [16] devised an efficient RR set generation approach that decreases the sampling time for each RR set. Hao et al. [17] propose a novel and effective framework for popularity maximization, designed to address the challenges of advertising competition. In the last few years, deep learning-based methods have emerged as alternative solutions. Fan et al. [18] introduced a deep reinforcement learning framework that learns the graph representation and sequentially selects key nodes one at each step. Ling et al. [4] developed a novel framework to generate latent representations of node sets and determine the

best node set only in a single step, in which the complex interactions among key nodes are encoded in the latent representations. Besides, there exist some other IM variants, such as time-critical IM [19], robustness IM [20, 21], online IM [22], and so on [23, 24].

*Group-level influence maximization.* Group-level influence maximization is an emerging field within social network analysis [9, 25, 11]. Groups refer to communities, cliques, and motifs, other kinds of subgraphs. The group-level influence maximization is to pinpoint influential nodes capable of triggering the most groups. Zhu et al. [9, 26, 27] have introduced the problem aimed at selecting key nodes to maximize the number of activated groups. They have also proven the NP-hardness of this problem and provided upper and lower bounds for the objective function. However, it is important to note that the lower bounds are dependent on the graph structure. Nguyen et al. [28] have proposed a method for identifying key nodes that exert influence over the largest number of communities. Zhong et al. [29, 30] have developed a heuristic approach to maximize the number of activated groups and have empirically demonstrated its effectiveness. Phuong et al. [31, 32] have minimized the cost of group influence maximization in social networks. However, we still lack an effective method to guarantee the approximation ratio of motif influence maximization.

Our study draws parallels with the group influence optimization in references [9] and [31]. However, our work stands out from prior studies through the following innovative contributions: (a) We first propose the motif-oriented influence maximization under the linear threshold model and explore the unique properties of this problem, which differ from traditional groups in a graph. (b) We propose both upper and lower bounds for the motif-oriented objective function. The two bounds share the same formalisms and can be optimized simultaneously, unlike previous solutions of group-oriented *IM* where the two bounds are different and cannot achieve the optimal solution simultaneously. (c) In contrast to previous heuristic approaches that cannot guarantee performance, we present a rapid algorithm that guarantees an approximation ratio. Therefore, our paper provides a viable approach to maximize motif-oriented influence in large graphs.

## 3   Problem definition

Let $G = (V, E)$ be a directed graph $G$ with $n$ nodes and $m$ edges, where a node $v \in V$ represents a user and an edge $(u, v) \in E$ represents the connection between users. $(u, v) \in E$ means that there is a directed edge from node $u$ to $v$. Each edge $(u, v) \in E$ is associated with a weight $p(u, v)$. Given a subset of nodes $S \in V$, denoted as initial activated seeds, we consider the following cascade process $\mathbb{C}$ that allows for both the Linear Threshold (LT) and Independent Cascade (IC) models:

- Initially, all nodes in set $S$ are activated and the other nodes are inactive.
- At each time step, when a node is activated, it has the opportunity to activate its out-neighbors in the following time step: Each node can activate its out-neighbors only once; Once a node is activated, it will remain active in the subsequent time steps.
- The cascade process terminates when there are no further nodes that can be activated.

Let $I_C(S)$ be the number of activated nodes for an instance $C$ of the above cascade process $\mathbb{C}$ on condition that $S$ is the initial seed node set. We denote $\mathbb{I}_{\mathbb{C}}(S) = \mathbb{E}_{\mathbb{C}}[I_C(S)]$ as the expected influence of $S$ under $\mathbb{C}$. The traditional IM aims to find the best $S$ to maximize $\mathbb{I}_{\mathbb{C}}(S)$. Here, we use motifs as the targeted customers for advertisements. We define a motif as a strongly connected subgraph:

**Definition 1(Motif definition)**. Given a subset of nodes $g$ in a graph, if there exist mutual paths between any pair of nodes in $g$, such that the paths only traverse the nodes within $g$, we denote $g$ as a motif(strongly connected group).

Roughly speaking, a motif refers to a small, strongly connected set of nodes, distinct from the strongly connected giant component of a graph. Motifs are commonly regarded as the fundamental functional units within graphs, a concept widely studied in the field of network science [? 8]. In order to facilitate group decision-making, it is essential that nodes within a motif have mutual accessibility, i.e., strongly connected. This requirement aligns well with various real-life scenarios. Establishing an acquaintance chain becomes indispensable to coordinate the activity occurring within the group.

Let $\mathfrak{g} = \{g_1, g_2, ..., g_h\}$ be the targeted motifs set. Let $\mu(v) = 1(0)$ denote whether node $v$ is activated or not and $\mu(g_i, r_i) = 1(0)$ denote whether motif $g_i$ is activated or not. Specifically, $\mu(g_i, r_i) = 1$ indicates that there are more than $r_i$ activated nodes within motif $g_i$ (i.e., the motif

is activated), while $\mu(g_i, r_i) = 0$ indicates that the number of activated nodes in motif $g_i$ is less than $r_i$. Now, let's define $I_C^{g'}(S)$ as the number of activated motifs resulting from an instance $C$, given that $S$ represents the initial seed nodes, $I_C^{g'}(S) = \sum_{i=1}^{h} \mu(g_i, r_i)$. In viral marketing, motifs of different sizes bring about different profits, hence a general motif-oriented influence function is the weighted number of activated motifs, $I_C^{g}(S) = \sum_{i=1}^{h} w_i \mu(g_i, r_i)$, where $w_i \geq 0$ is the weight of motif $i$. Additionally, we denote $\mathbb{I}_{\mathbb{C}}^{g}(S) = \mathbb{E}_{\mathbb{C}}[I_C^{g}(S)]$ as the expected weighted influence of seed nodes $S$ under the cascade process $\mathbb{C}$. Hence the problem in the paper is formalized as:

**Problem Definition 1(Motif-oriented influence maximization)**. Given a graph $G$, a cascade model $\mathbb{C}$, and an integer $k$, the **M**otif-**O**riented **I**nfluence **M**aximization (MOIM) asks for a size-$k$ seed set with the largest expected activated motifs, i.e.,

$$S_k = argmax_{S:|S|=k} \mathbb{I}_{\mathbb{C}}^{g}(S). \tag{1}$$

**Cascade Models**: Our paper primarily focuses on the analysis of the LT model. The LT model requires that $\sum_u p(u,v) \leq 1$. In the LT model, each node is associated with a uniformly random threshold $\lambda_v \in [0,1]$. When a node $v$ is inactive at time $t$, it will become active at time $t+1$ on condition that $\sum_{u \in A_v} p(u,v) \cdot \mu(u) \geq \lambda_v$, where $A_v$ represents the activated in-neighbors of node $v$. Notably, our findings are not limited to the LT model and can be extended to the IC model and the triggering model, which will be discussed in Section 5.

# 4 Complexity analysis of MOIM

**Theorem 1.** The MOIM problem is NP-hard for the linear threshold model.

*Proof.* Given that the classical IM problem can be treated as a particular case of MOIM where each motif consists of a single node, since the classical IM problem is NP-hard [3], MOIM is also an NP-hard problem. Strict proof details are shown in the appendix C. □

**Theorem 2.** The MOIM problem is neither submodular nor supermodular.

*Proof.* We present two illustrative examples to demonstrate the non-submodular and non-supermodular properties. The submodular property states that the marginal gain of adding a node to a set of more seed nodes should decrease. More formally, for any two sets of nodes $S, T(S \subseteq T)$, and a node $v(v \notin T)$, the inequality $\mathbb{I}_{\mathbb{C}}(S \cup \{v\}) - \mathbb{I}_{\mathbb{C}}(S) \geq \mathbb{I}_{\mathbb{C}}(T \cup \{v\}) - \mathbb{I}_{\mathbb{C}}(T)$ should hold.

Case 1: We consider a graph with four nodes and a target motif as shown in Fig. 1(a). When the seed nodes are $S_1 = \{1\}$ (or $\{2\}$), the expected activated motif is $\mathbb{I}_{\mathbb{C}}(S_1) = p(1-p)$. On the other hand, when the seed nodes are $T = \{1,2\}$, the expected activated motif is $\mathbb{I}_{\mathbb{C}}(T) = 2p(1-p) + (1-p)^3$. As $p \to 0$, we observe that $\mathbb{I}_{\mathbb{C}}(\emptyset \cup \{1\}) - \mathbb{I}_{\mathbb{C}}(\emptyset) = \mathbb{I}_{\mathbb{C}}(S_1) < \mathbb{I}_{\mathbb{C}}(\{1,2\}) - \mathbb{I}_{\mathbb{C}}(\{2\})$. Consequently, the MOIM model fails to satisfy the submodular property.

Case 2: We examine the MOIM problem under the specific scenario where a node represents a motif. In this case, the MOIM problem is degenerated into the classical IM problem. Notably, the marginal gain of seed nodes in the IM problem satisfies the submodular property. However, as a result of this reduction, the general MOIM model no longer adheres to the supermodular property.

Combining the two cases, we arrive at the theorem. □

# 5 The proposed solution

## 5.1 The upper and lower bounds of the objective function

Given that the objective function of $\mathbb{I}_{\mathbb{C}}^{g}(S)$ does not exhibit submodular or supermodular properties, a feasible approach is to optimize the upper and lower bounds of $\mathbb{I}_{\mathbb{C}}^{g}(S)$, i.e., the sandwich strategy. The central issue of the sandwich strategy is to obtain the upper and lower bounds. Thus, in the subsequent sections, we consider two cases and derive the upper and lower bounds of $\mathbb{I}_{\mathbb{C}}^{g}(S)$.

Case 1 ($r_i = 1$): We first consider the IM example in Fig. 1(a) and the scenario where $r_i = 1, \forall i = 1, 2, ..., h$. In this case, a motif will be activated if at least one node within the motif is activated. To account for this, we introduce a super node for each motif, as shown in Fig. 1(b). Every node within

a motif is then connected to its corresponding super node with an activation probability of $p = 1$ in Fig. 1(b). The activation of the super node follows the independent cascade model, meaning that the super node is activated if at least one node within the motif is activated. Let $T$ represent the set

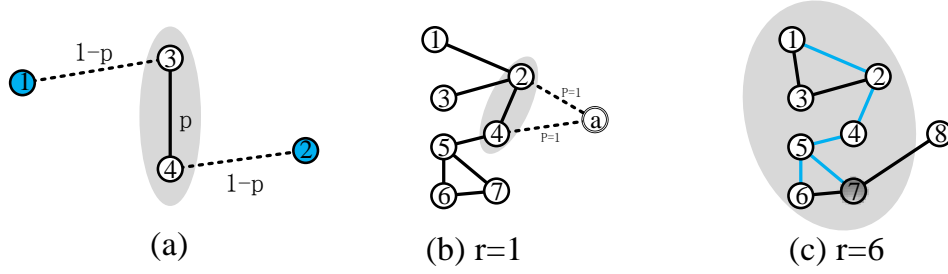

Figure 1: (a) An example of a graph containing only one motif. Nodes 1 and 2 are the candidate seed nodes. (b) A super node example(labeled $a$) for a motif, which connects to two nodes labeled 2 and 4. (c) 5-spanning tree example (see blue edges).

of super nodes, $|T| = h$, and define $\bar{f}(S) = \mathbb{E}_{\mathbb{C}}[\sum_{a \in T} w_i \mu(a)]$ as the expected number of activated super nodes weighted by the respective weights $w_i$, given the initially activated seeds $S$ ($S \notin T$). The expectation of activated motifs can then be represented as $\mathbb{I}_{\mathbb{C}}{}^{\mathfrak{g}}(S) = \bar{f}(S)$.

**Remark 1**. When $r_i = 1$, the MOIM problem can be reformulated as the modified IM problem, which seeks to determine the optimal seed set $S$ for maximizing the activation of the designated super node set $T$. Besides, one node may connect to equal or more than two supper nodes.

Case 2 ($r_i > 1$): We divide the cascade process into two stages for the convenience of analysis: Given a motif $g_i$, the cascade first occurs along the edges outside the motif $g_i$, and then the cascade occurs between nodes within the motif $g_i$. Let $P_{out}(S, g_i, A)$ be the joint distribution of the activated node subset $A$ ($A \subseteq g_i, A \neq \emptyset$) in the cascade stage. In the first stage, when the number of activated nodes $|A|$ is less than $r_i$, in order to ensure the activation of motif $g_i$, there must be at least $r_i - |A|$ active edges connecting the activated nodes in $A$ and the inactive nodes in $g_i \setminus A$.

We define a "semi $d$-spanning tree starting from node set $A$" as a subset of edges within the motif $g_i$ that connect nodes in $A$ and $d$ other nodes in $g_i$ without forming any cycles, denoted as $E_d(A)$. Fig. 1(c) illustrates an example of a semi 5-spanning tree starting from the node set $A = \{7\}$. Let $P(E_d(A)) = \prod_{(u,v) \in E_d(A)} p(u, v)$, and $P(E_d(A))$ represents the realization probability that the edges in set $E_d(A)$ are concurrently activated in the cascade. If $E_d(A)$ is empty, we then set $P(E_d(A)) = 1$. Let $P(E_d^{(i)})_{min}$ represent the minimal value of $P(E_d(A))$ for semi $d$-spanning tree in the motif $g_i$. The activation probability of motif $g_i$ is given by:

$$Pr[\mu(g_i, r_i) = 1] = \sum_{A \neq \emptyset} \sum_{d, d \geq r_i - |A|} P_{out}(S, g_i, A) \cdot P(E_d(A))$$

$$\geq \sum_{A \neq \emptyset} \sum_{d, d \geq r_i - |A|} P_{out}(S, g_i, A) \cdot P(E_d^{(i)})_{min}. \tag{2}$$

Since $A \neq \emptyset$ and $d \leq r_i - 1$, we have $P(E_d^{(i)})_{min} \geq P(E_{r_i-1}^{(i)})_{min}$. Additionally, it is worth noting that $Pr[\mu(g_i, 1) = 1] = \sum_{A \neq \emptyset} P_{out}(S, g_i, A)$ represents the probability that at least one node in motif $g_i$ is activated by the outside nodes of $g_i$. Thus, we have:

$$Pr[\mu(g_i, r_i) = 1] \geq \sum_{A \neq \emptyset} \sum_{d, d \geq r_i - |A|} P_{out}(S, g_i, A) \cdot P(E_{r_i-1}^{(i)})_{min}$$

$$\geq P(E_{r_i-1}^{(i)})_{min} \cdot Pr[\mu(g_i, 1) = 1]. \tag{3}$$

Recalling that $\bar{f}(S) = \mathbb{E}_{\mathbb{C}}[\sum_{a \in T} w_i \mu(a)]$ represents the expected number of activated super nodes weighted by the weights $w_i, i = 1, 2, ..., h$. let $w_i' = w_i \cdot P(E_{r_i-1}^{(i)})_{min}, \tau = min\{P(E_{r_i-1}^{(i)})_{min}, i = 1, 2, ..., h\}, \underline{f}_1(S) = \mathbb{E}_{\mathbb{C}}[\sum_{a \in T} w_i' \mu(a)]$, and $\underline{f}_2(S) = \tau \cdot \bar{f}(S)$. we have the following theorem:

**Theorem 3(Upper and lower bounds)**. $\bar{f}(S)$ is the upper bound of $\mathbb{I}_{\mathbb{C}}{}^{\mathfrak{g}}(S)$; $\underline{f}_1(S)$ and $\underline{f}_2(S)$ are two lower bounds of $\mathbb{I}_{\mathbb{C}}{}^{\mathfrak{g}}(S)$, $\underline{f}_2(S) \leq \underline{f}_1(S) \leq \mathbb{I}_{\mathbb{C}}{}^{\mathfrak{g}}(S) \leq \bar{f}(S)$. Besides, $\underline{f}_2(S) = \tau \cdot \bar{f}(S)$.

*Proof*: Since $Pr[\mu(g_i, r_i)] \leq Pr[\mu(g_i, 1)]$ and $w_i \in [0,1]$, $\mathbb{I}_{\mathbb{C}}{}^{\mathfrak{g}}(S) = \mathbb{E}_{\mathbb{C}}[\sum_i w_i \mu(g_i, r_i)] \leq \mathbb{E}_{\mathbb{C}}[\sum_i w_i \mu(g_i, 1)] = \bar{f}(S)$. Hence, $\bar{f}(S)$ is the upper bound of $\mathbb{I}_{\mathbb{C}}{}^{\mathfrak{g}}(S)$.

Based on Eq. 3, $\mathbb{I}_{\mathbb{C}}{}^{\mathfrak{g}}(S) = \mathbb{E}_{\mathbb{C}}[\sum_i w_i \mu(g_i, r_i)] \geq \sum_i w_i (P(E_{r_i-1}^{(i)})_{min} \cdot Pr[\mu(g_i, 1) = 1]) \geq \underline{f}_1(S) \geq \sum_i w_i (\tau \cdot Pr[\mu(g_i, 1) = 1]) = \underline{f}_2(S).\square$

## 5.2 The proposed algorithm LBMOIM

It is noticed that $\bar{f}(S)$, $\underline{f}_1(S)$, and $\underline{f}_2(S)$ share the similar formalism. In the section, we focus on optimizing $\bar{f}(S)$ to calculate the best seeds, which is also capable for $\underline{f}_1(S)$ and $\underline{f}_2(S)$.

**Node selection based on RR set:** In Algorithm 1, we determine the optimal seeds by optimizing the upper and lower bounds. Initially, a certain number $\theta$ of RR sets are generated (lines 2-6), followed by the standard greedy algorithm to generate a set $S_k^*$ of size-$k$ nodes that covers the maximum number of RR sets in $\mathcal{R}$ (lines 7-10). The method is similar to the node-level IM solution (see Appendix A). The only difference is the probability of choosing root nodes in the generation of RR set. The key problem of Algorithm 1 is determining the number $\theta$. Let $\mathcal{R}$ denote the set of all RR sets generated by randomly selecting a super node $v$ with a probability of $\frac{w_v}{\eta}$, where $\eta = \sum_j w_j$. Let $F_R(S)$ be the fraction of RR sets in $\mathcal{R}$ covered by $S$. We have the following lemma:

**Lemma 1**. Assume that $\theta$ follows

$$\theta \geq (8 + 2\varepsilon)\eta \cdot \frac{l \log \eta + \log \binom{\eta}{k} + log2}{OPT \cdot \varepsilon^2}. \tag{4}$$

Then, for any size-$k$ set of seeds, the following inequality holds with at least probability $1 - \eta^l / \binom{\eta}{k}$:

$$|\eta \cdot F_{\mathcal{R}}(S) - \bar{f}(S)| < \frac{\varepsilon}{2} \cdot OPT, \tag{5}$$

where $OPT$ means the best-activated motif expectation of size-$k$ seeds.

*Proof*: Comparing to Eq. 7, the only difference is replacing $n$ by $\eta$ in Eq. 4 because we have $\eta$ weighted target motifs in our task. Since the modification doesn't influence the proof derivation in reference [14], please see reference [14] for the details. $\square$

**Theorem 4 [14]**. Given a $\theta$ that satisfies Eq. 4, Algorithm 1 returns a $(1 - 1/e - \varepsilon)$-approximate solution with at least probability $1 - \eta^{-l}$.

*Proof*: Please see reference [14] for the details. $\square$

**Parameter estimation**: The major problem lies in the prior evaluation of $OPT$. To differentiate it from classical IM, we denote the optimal value as $OPT_{\bar{f}} = max\{\bar{f}(S), |S| = k\}$. In this part, we evaluate the parameters $\theta$ and $OPT_{\bar{f}}$. We generalize the estimation of $OPT$ in TIM [14] to weighted influence function to estimate a lower bound of $OPT_{\bar{f}}$, denoted as $KPT$ (see Algorithm 3 in appendix B). We could estimate $KPT$ in Algorithm 3 $KPT^* \in [KPT/4, OPT_{\bar{f}}]$ with a probability of at least $1 - \eta^{-l}$ and expected time complexity of $O(l(m + \eta)log\eta)$ (see the proof in appendix B).

We present Algorithm 2, which optimizes the **L**ower **B**ound of MOIM objective function (LBMOIM) to calculate the best nodes. In Algorithm 2, we initially assess the value of $\theta$ (lines 1–2) and subsequently compute the best nodes (line 3). Optimizing the lower bound $\underline{f}_2(S)$ also is equivalent to the optimization of the upper bound $\bar{f}(S)$, and hence we do not differentiate them and omit them.

**Lemma 2(time complexity)**. The time complexity of Algorithm 2 is $O((k + l)(m + \eta) \log \eta/\varepsilon^2)$.

*Proof*: Please see the appendix C for the proof details. $\square$

**Lemma 3(Approximation confidence)**. In Algorithm 2, the $S^*$ is $(1-1/e-\varepsilon)$-approximate solution with a probability of at least $1 - 2 \cdot \eta^{-l}$.

*Proof:* Given that the objective function $\bar{f}(S)$ is a weighted version of the IM problem that satisfies the submodular property, the greedy algorithm is capable of producing a $(1 - 1/e - \varepsilon)$-approximate

**Algorithm 1:** NodeSelection $(G, \theta, k)$.

---

1 Initialize a node set $S_k^* = \emptyset$;
2 **for** $i = 1 : \theta$ **do**
3     Randomly choose a super node $v$ with probability $\frac{w_v}{\eta}$;
4     Generate a RR set $R_i$ starting from node $v$;
5     Insert $R_i$ to $\mathcal{R}$;
6 **end**
7 **for** $i = 1 : k$ **do**
8     Identify the node $v$ that maximizes the marginal coverage of $\mathcal{R}$, $F_R(S_k^* \cup \{v\}) - F_R(S_k^*)$;
9     Insert $v$ into $S_k^*$;
10 **end**
11 return $\mathcal{S}_k^*$;

---

solution for the submodular function. Additionally, by combining Eq. 5 with Theorem 4, we can deduce the probability of at least $1 - 2 \cdot \eta^{-l}$.

**Lemma 4(Approximation ratio).** Let $S^*$ be the optimal solution that maximizes $\mathbb{I}_\mathbb{C}{}^\mathfrak{g}(S)$, and $\bar{S}^*$ be the solution returned by Algorithm 2 that maximizes $\bar{f}(S)$. Algorithm 3 achieves an approximation ratio $\gamma = \frac{f(\bar{S}^*)}{\mathbb{I}_\mathbb{C}{}^\mathfrak{g}(S^*)} \geq \tau \cdot (1 - 1/e - \varepsilon)$.

*Proof:* Based on Theorem 3, we have $\tau \cdot \bar{f}(S^*) \leq \mathbb{I}_\mathbb{C}{}^\mathfrak{g}(S^*) \leq \bar{f}(S^*)$. Furthermore, based on Lemma 3, $\bar{S}^*$ can achieve a $(1 - 1/e - \varepsilon)$ approximation ratio. Therefore, we can deduce that $\mathbb{I}_\mathbb{C}{}^\mathfrak{g}(S^*) \leq \bar{f}(S^*) \leq \bar{f}(\bar{S}^*)/(1 - 1/e - \varepsilon)$. The approximation ratio $\gamma = \frac{f(\bar{S}^*)}{\mathbb{I}_\mathbb{C}{}^\mathfrak{g}(S^*)} \geq \frac{\tau \cdot \bar{f}(\bar{S}^*)}{\bar{f}(\bar{S}^*)/(1 - 1/e - \varepsilon)} = \tau \cdot (1 - 1/e - \varepsilon)$. $\square$

---

**Algorithm 2:** (LBMOIM)summarization $(G, k)$.

---

1 $KPT = KptEstimation(G, k)_{\bar{f}}$;
2 $\theta = (8 + 2\varepsilon)\eta \cdot \frac{l \log \eta + \log \binom{\eta}{k} + log2}{KPT \cdot \varepsilon^2}$;
3 $S^* = NodeSelection(G, \theta, k)$;
4 return $S^*$;

---

**Improved MOIM solution**: In the optimization, we only optimize $\bar{f}(S)$ ($\underline{f}_2(S)$). For the optimization of $\underline{f}_1(S)$, we could use Algorithms 1–3 to calculate the best node set that maximizes $\underline{f}_1(S)$ with minor modification: replacing $w_i$ and $\eta = \sum_i w_i$ with $w_i'$ and $\eta' = \sum_i w_i'$ in Algorithms 1–3. Suppose that $\underline{S}_1^*$ and $\underline{S}_2^*$ are the returns of Algorithm 3 and correspond to the best $\underline{f}_1(S)$ and $\underline{f}_2(S)$. Let $\underline{S}^* = argmax_S\{f(S)|S \in \{\underline{S}_1^*, \underline{S}_2^*\}\}$. We bear the conception that $\underline{S}^*$ performs better(at least equal to) $\underline{S}_2^*$. Besides, calculating $\underline{S}^*$ has the same time complexity and approximation ratio, which is omitted due to space limitation.

**Remark 2**: Our algorithm could also be generalized to the independent cascade (IC) model and triggering model [3]. Please see Appendix D for the generalization.

## 6 Experiments

### 6.1 Experimental setup

**Datasets.** We use five real social networks in Table 1 in our experiments. All datasets are available in KonectCollection [2].

**Baseline methods.** We perform a comparative analysis of our proposed method against five existing algorithms: TIM+$^{SIGMOD2014}$ [14], OPIM$^{ICMD2018}$ [15], DeepIM$^{ICML2023}$ [4], GIM$^{TCSS2019}$ [9], and GIA$^{CC2023}$ [31]. TIM, OPIM, and IMM are among the most advanced techniques in the

Table 1: Dataset characteristics.

| Name | $n$ | $m$ | Type | Average degree |
|---|---|---|---|---|
| Flickr | $2M$ | $33.1M$ | directed | 28.8 |
| Amazon | $334K$ | $925.8K$ | undirected | 5.5 |
| Catster | $623K$ | $14.0M$ | undirected | 44.9 |
| Youtube | $1M$ | $3.0M$ | undirected | 5.3 |
| Douban | $154K$ | $327.2K$ | undirected | 4.2 |

node-level IM problem, with a high approximation ratio $(1 - 1/e - \varepsilon)$. On the other hand, GIM and GIA are specifically designed to address the MOIM problem under the IC model. While GIM and EGI achieve a similar approximation ratio $(1 - 1/e - \varepsilon)$ in terms of the objective functions, the objective functions are slightly different from the targeted influence function.

**Parameter Settings.** For our method, we set the default parameters as follows: $\varepsilon = 0.3$, $l = 1$, $w_i = 1$ by default. We randomly choose 2000 strongly connected subgraphs as target motifs with motif size being 3 unless otherwise stated. As for the other methods, we have employed the recommended parameters from their original references. Two cascade models, namely the LT and IC models, have been considered. To ensure reliability, each method has been executed 10 times, and the average results have been reported. Furthermore, we have estimated the expected influence of any given seed set $S$ by averaging the results from 100 independent cascade simulations. Our experiments run on a computer with one i7 CPU, 32GB memory and C++ development environment. The code is available at https://anonymous.4open.science/r/motifinfluencemax-7863.

**Evaluation metrics.** The primary criterion for evaluation is the motif influence function $\mathbb{I}_{\mathbb{C}}(S)$, and larger $\mathbb{I}_{\mathbb{C}}(S)$ is better. The second metric is the time cost. The secondary criterion pertains to the time incurred during the process. In the case where different methods exhibit identical $\mathbb{I}_{\mathbb{C}}(S)$ values, the preference is given to the method with a smaller time cost.

## 6.2 Experiment results

**Results under the LT model**: In Fig. 2, we present the performance of the motif influence function under the LT model. The figure shows that our method LBMOIM surpasses existing methods across various graphs. Additionally, comparing node-level methods (TIM+, OPIM, DeepIM) with motif-targeted methods (GIM, GIA, LBMOIM), we find that the latter outperforms the former due to a slight difference in the target objective function. Notably, for our method LBMOIM, we use a simplified objective function that ensures both the approximation ratio and submodular property, resulting in the best performance (Fig. 2). Furthermore, we consider the scenario of triangle subgraphs as motifs and setting $r_i = 1$. Fig. 3 shows the number of activated triangle motifs with a motif size of 3 and $r_i = 1$. Once again, the results reaffirm the superiority of our method, aligning with the findings from Fig. 2. Additionally, we compare the number of activated motifs performance at $r_i = 2$ in Fig. 4(a) and at $r_i = 3$ in Fig. 4(b). In Fig. 4(c), we activate half of the motifs at $r_i = 2$ and the other half at $r_i = 3$. In this scenario, we optimize the lower bound and upper bound of the motif influence function. Considering only motif size 3, $\underline{f}_1(S)$ degenerates into $\underline{f}_2(S)$, and therefore, we do not differentiate between the two lower bounds. Fig. 4 further confirms that our method outperforms existing methods.

In Figs. 2 and 3, we have investigated the performance under different motif size. We have also performed the case with a larger motif size. The results show that the performances of all methods deteriorate with the increase of the motif size. However, our method still achieves the best performance. Since the results are similar to Figs. 2 and 3, we omit them due to space limitation.

**Results under the IC model:** Our proposed method can be extended to the IC model and triggering model. Fig. 5 illustrates the motif coverage performance under the IC model. Specifically, we consider threesome motifs and set $r_i$ values equal to 1, 2, and 3 using the Douban dataset. Fig. 5 clearly demonstrates that our method consistently outperforms existing methods. It is important to note that the GIA method exhibits a similar performance to our method in Fig. 5(b). However, it is worth mentioning that GIA lacks an approximation ratio guarantee, resulting in fluctuating performance as shown in Figs. 5(a) and 5(c). Furthermore, it is important to highlight that node-level

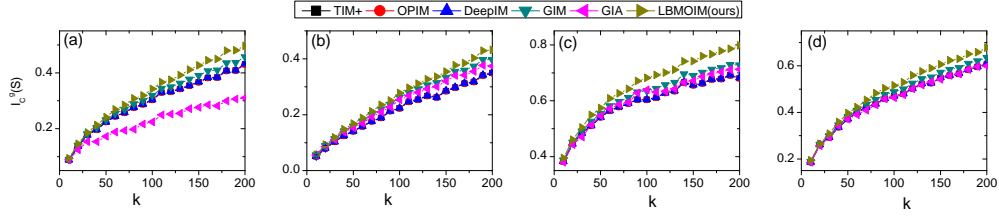

Figure 2: Expected motif influence vs. $k$ under LT model, $r_i = 1$ and motif size being 2. (a) Flickr. (b) Amazon. (c) Catster. (d) Youtube. Larger is better.

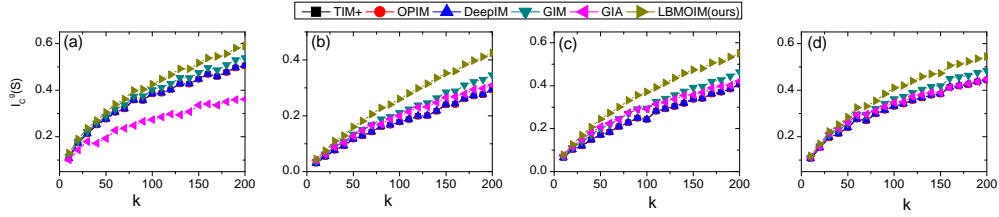

Figure 3: Expected motif influence vs. $k$ under LT model, $r_i = 1$ and motif size being 3. (a) Flickr. (b) Amazon. (c) Catster. (d) Youtube. Larger is better.

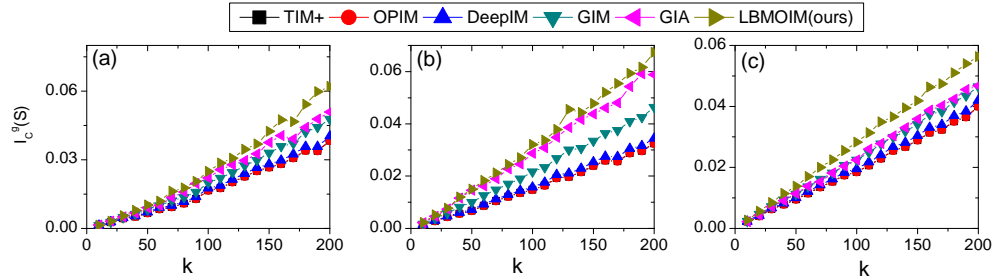

Figure 4: Expected motif influence vs. $k$ under LT model. (a) Flickr. Motif size is 2 and $r_i = 2$. (b) Amazon. Motif size is 3 and $r_i = 3$. (c) Catster. Half motifs have size 2 and $r_i = 2$, whereas the other half motifs have size 3 and $r_i = 3$. Larger is better.

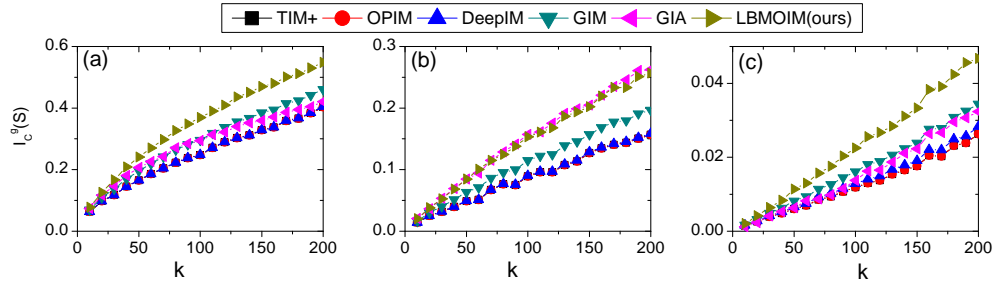

Figure 5: Expected motif influence vs. $k$ under IC model in Douban dataset, $r_i = 1$ and motif size being 3. (a) $r_i = 1$. (b) $r_i = 2$. (c) $r_i = 3$. Larger is better.

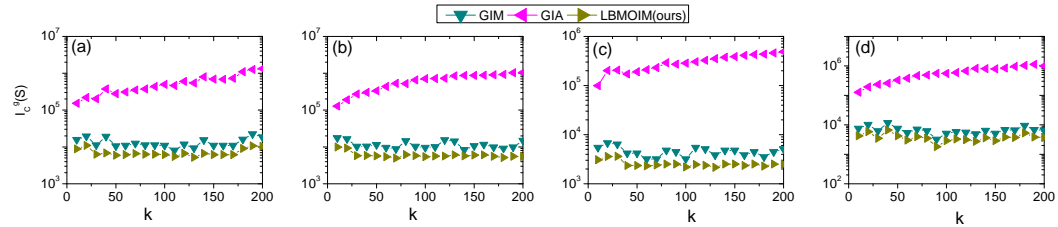

Figure 6: (a) Expected time consumption(milliseconds) vs. $k$ under LT model, $r_i = 1$ and motif size being 3. (a) Flickr. (b) Amazon. (c) Catster. (d) Youtube. Smaller is better.

methods such as TIM+, OPIM, and DeepIM, which have different target objective functions, also cannot achieve the best performance under the IC model.

**Results of time consumption:** The time consumption of different approaches is compared in Fig. 6. Specifically, the comparison focuses on GIM, GIA, and LBMOIM, as these methods exclusively optimize motif coverage. GIM exhibits the highest time consumption due to the need to optimize two node-level objective functions, in contrast to the other two methods which only optimizes a single objective function. Importantly, it should be noted that the time consumption of our proposed method exhibits a minor increase with respect to $k$, demonstrating its scalability when applied to large graphs. This observation aligns with the time complexity analysis in Lemma 2.

# 7    Conclusion

In this paper, we have addressed the motif-oriented influence maximization (MOIM) problem. To tackle the non-supermodular and non-submodular MOIM objective function, we have derived simplified submodular upper and lower bounds that make the problem more tractable. We further propose an efficient algorithm capable of optimizing these bounds. The algorithm runs in expected time of $O((k + l)(m + \eta) \log \eta / \varepsilon^2)$ and achieves an approximation ratio of $\tau \cdot (1 - 1/e - \varepsilon)$. We experimentally evaluate our method against the state-of-the-art methods on large real-world social networks using the LT and IC models. Our results consistently demonstrate that our method outperforms state-of-the-art methods in terms of both motif influence and computational efficiency. These findings highlight the potential value of motif-oriented approaches in viral marketing and social influence scenarios.

In future work, we plan to delve deeper into the intricate structural properties of the motif influence function, aiming to uncover new insights into its mathematical behavior and relationships. This includes exploring more refined and tighter bounds that can guide the development of algorithms with improved performance guarantees. Additionally, we intend to extend our investigation to handle more complex motifs across diverse and heterogeneous network structures, where interactions are dynamic or context-dependent.

# 8    Acknowledgments

The authors thank Dr. Wei Chen for his invaluable guidance and insightful discussions, and the anonymous reviewers for their constructive feedback and insightful comments. The authors acknowledge the financial support from the National Natural Science Foundation of China (Grant Nos. 62476173, 62276171, 62002233, 61972145), the Shenzhen Fundamental Research-General Project (Grant Nos. JCYJ20240813142610014, JCYJ20240813141503005, JCYJ20220811155803001), Guangdong Basic and Applied Basic Research Foundation (Grant Nos. 2024A1515011938 and 2020B1515120028), Guangdong Peral River Recruitment Program of Talents (Grant Nos. 2019ZT08X603). Hao Liao is the corresponding author.

## Footnotes

[2]http://konect.cc/networks/

# References

[1] Yuchen Li, Ju Fan, Yanhao Wang, and Kian-Lee Tan. Influence maximization on social graphs: A survey. *IEEE Transactions on Knowledge and Data Engineering*, 30(10):1852–1872, 2018.

[2] Wei Chen, Chi Wang, and Yajun Wang. Scalable influence maximization for prevalent viral marketing in large-scale social networks. In *Proceedings of the 16th ACM SIGKDD international conference on Knowledge discovery and data mining*, pages 1029–1038, 2010.

[3] David Kempe, Jon Kleinberg, and Éva Tardos. Maximizing the spread of influence through a social network. In *Proceedings of the ninth ACM SIGKDD international conference on Knowledge discovery and data mining*, pages 137–146, 2003.

[4] Chen Ling, Junji Jiang, Junxiang Wang, My T Thai, Renhao Xue, James Song, Meikang Qiu, and Liang Zhao. Deep graph representation learning and optimization for influence maximization. In *International Conference on Machine Learning*, pages 21350–21361. PMLR, 2023.

[5] Yushun Dong, Jing Ma, Song Wang, Chen Chen, and Jundong Li. Fairness in graph mining: A survey. *IEEE Transactions on Knowledge and Data Engineering*, 2023.

[6] Rakesh R Mallipeddi, Subodha Kumar, Chelliah Sriskandarajah, and Yunxia Zhu. A framework for analyzing influencer marketing in social networks: selection and scheduling of influencers. *Management Science*, 68(1):75–104, 2022.

[7] Pedro Ribeiro, Pedro Paredes, Miguel EP Silva, David Aparicio, and Fernando Silva. A survey on subgraph counting: concepts, algorithms, and applications to network motifs and graphlets. *ACM Computing Surveys (CSUR)*, 54(2):1–36, 2021.

[8] Ron Milo, Shai Shen-Orr, Shalev Itzkovitz, Nadav Kashtan, Dmitri Chklovskii, and Uri Alon. Network motifs: simple building blocks of complex networks. *Science*, 298(5594):824–827, 2002.

[9] Jianming Zhu, Smita Ghosh, and Weili Wu. Group influence maximization problem in social networks. *IEEE Transactions on Computational Social Systems*, 6(6):1156–1164, 2019.

[10] Diego García-Zamora, Álvaro Labella, Weiping Ding, Rosa M Rodríguez, and Luis Martínez. Large-scale group decision making: a systematic review and a critical analysis. *IEEE/CAA Journal of Automatica Sinica*, 9(6):949–966, 2022.

[11] Mehdi Azaouzi, Wassim Mnasri, and Lotfi Ben Romdhane. New trends in influence maximization models. *Computer Science Review*, 40:100393, 2021.

[12] Youze Tang, Yanchen Shi, and Xiaokui Xiao. Influence maximization in near-linear time: A martingale approach. In *Proceedings of the 2015 ACM SIGMOD international conference on management of data*, pages 1539–1554, 2015.

[13] Hung T Nguyen, My T Thai, and Thang N Dinh. Stop-and-stare: Optimal sampling algorithms for viral marketing in billion-scale networks. In *Proceedings of the 2016 international conference on management of data*, pages 695–710, 2016.

[14] Youze Tang, Xiaokui Xiao, and Yanchen Shi. Influence maximization: Near-optimal time complexity meets practical efficiency. In *Proceedings of the 2014 ACM SIGMOD international conference on Management of data*, pages 75–86, 2014.

[15] Jing Tang, Xueyan Tang, Xiaokui Xiao, and Junsong Yuan. Online processing algorithms for influence maximization. In *Proceedings of the 2018 International Conference on Management of Data*, pages 991–1005, 2018.

[16] Qintian Guo, Sibo Wang, Zhewei Wei, and Ming Chen. Influence maximization revisited: Efficient reverse reachable set generation with bound tightened. In *Proceedings of the 2020 ACM SIGMOD international conference on management of data*, pages 2167–2181, 2020.

[17] Hao Liao, Sheng Bi, Jiao Wu, Wei Zhang, Mingyang Zhou, Rui Mao, and Wei Chen. Popularity ratio maximization: Surpassing competitors through influence propagation. *Proc. ACM Manag. Data*, 1(2), June 2023.

[18] Changjun Fan, Li Zeng, Yizhou Sun, and Yang-Yu Liu. Finding key players in complex networks through deep reinforcement learning. *Nature machine intelligence*, 2(6):317–324, 2020.

[19] Wei Chen, Wei Lu, and Ning Zhang. Time-critical influence maximization in social networks with time-delayed diffusion process. In *Proceedings of the AAAI Conference on Artificial Intelligence*, volume 26, pages 591–598, 2012.

[20] Meghna Lowalekar, Pradeep Varakantham, and Akshat Kumar. Robust influence maximization. 2016.

[21] Wei Chen, Tian Lin, Zihan Tan, Mingfei Zhao, and Xuren Zhou. Robust influence maximization. In *Proceedings of the 22nd ACM SIGKDD international conference on Knowledge discovery and data mining*, pages 795–804, 2016.

[22] Shuai Li, Fang Kong, Kejie Tang, Qizhi Li, and Wei Chen. Online influence maximization under linear threshold model. *Advances in Neural Information Processing Systems*, 33:1192–1204, 2020.

[23] Suman Banerjee, Mamata Jenamani, and Dilip Kumar Pratihar. A survey on influence maximization in a social network. *Knowledge and Information Systems*, 62:3417–3455, 2020.

[24] Yandi Li, Haobo Gao, Yunxuan Gao, Jianxiong Guo, and Weili Wu. A survey on influence maximization: From an ml-based combinatorial optimization. *ACM Transactions on Knowledge Discovery from Data*, 17(9):1–50, 2023.

[25] Alessia Antelmi, Gennaro Cordasco, Carmine Spagnuolo, and Przemysław Szufel. Social influence maximization in hypergraphs. *Entropy*, 23(7):796, 2021.

[26] Jianming Zhu, Junlei Zhu, Smita Ghosh, Weili Wu, and Jing Yuan. Social influence maximization in hypergraph in social networks. *IEEE Transactions on Network Science and Engineering*, 6(4):801–811, 2018.

[27] Jianming Zhu, Smita Ghosh, Weili Wu, and Chuangen Gao. Non-submodular model for group profit maximization problem in social networks. *Computational Social Networks*, 8(1):1–18, 2021.

[28] Lan N Nguyen, Kunxiao Zhou, and My T Thai. Influence maximization at community level: A new challenge with non-submodularity. In *2019 IEEE 39th International Conference on Distributed Computing Systems (ICDCS)*, pages 327–337. IEEE, 2019.

[29] Yuting Zhong, Longkun Guo, and Peihuang Huang. Maximizing group coverage in social networks. In *21st International Conference on Parallel and Distributed Computing, Applications and Technologies*, pages 274–284. Springer, 2021.

[30] Yuting Zhong and Longkun Guo. Group influence maximization in social networks. In *International Conference on Computational Data and Social Networks*, pages 152–163. Springer, 2020.

[31] Phuong NH Pham, Canh V Pham, Hieu V Duong, Václav Snášel, and Nguyen Trung Thanh. Minimizing cost for influencing target groups in social network: A model and algorithmic approach. *Computer Communications*, 212:182–197, 2023.

[32] Phuong NH Pham, Canh V Pham, Hieu V Duong, Trung Thanh Nguyen, and My T Thai. Groups influence with minimum cost in social networks. In *Computational Data and Social Networks: 10th International Conference, CSoNet 2021, Virtual Event, November 15–17, 2021, Proceedings 10*, pages 231–242. Springer, 2021.

[33] Christian Borgs, Michael Brautbar, Jennifer Chayes, and Brendan Lucier. Maximizing social influence in nearly optimal time. In *Proceedings of the twenty-fifth annual ACM-SIAM symposium on Discrete algorithms*, pages 946–957. SIAM, 2014.

## A    Existing node-level IM

Most existing scalable IM solutions consist of two phases:

- *Sampling*. This phase iteratively generates a certain number of random reverse reachable(RR) sets $\mathcal{R}_i$, denoted as $\mathcal{R} = \{\mathcal{R}_1, \mathcal{R}_2, ..., \mathcal{R}_\theta\}$.

- *Node selection*. This phase utilizes the standard greedy algorithm to derive a size-$k$ node set $S_k^*$ that covers the most number of RR sets in $\mathcal{R}$. The $S_k^*$ is the output of the best node seeds.

In the sampling phase, an RR set is constructed through two steps: Firstly, the direction of each edge in $E$ is reversed. Secondly, a node $v$ is randomly selected from $V$, and an instance of the cascade is obtained by setting $v$ as the initial activated node using the cascade model $\mathbb{C}$. The resulting set of activated nodes is denoted as an RR set. The RR set exhibits the following desirable properties:

**Lemma 5** [33]. Consider a seed set $S$ and a randomly generated RR set $\mathcal{R}_i$ under the diffusion model $\mathbb{C}$. The expected influence $\mathbb{I}_\mathbb{C}(S)$ can be expressed as:

$$\mathbb{I}_\mathbb{C}(S) = n \cdot Pr[S \cap \mathcal{R}_i \neq \emptyset]. \tag{6}$$

Borgs et al. [33] initially examine the efficiency of the greedy algorithm based on the aforementioned two phases. Tang et al. [14] introduce an improved algorithm called TIM, which has a time complexity of $O((k + l)(n + m) \log n / \varepsilon^2)$. TIM estimates $\theta$ as follows:

$$\theta \geq (8 + 2\varepsilon)n \cdot \frac{l \log n + \log \binom{n}{k} + log2}{OPT \cdot \varepsilon^2}, \tag{7}$$

where $l$ and $OPT$ represent a factor coefficient and the best influence of size-$k$ node set, respectively. Let $F_\mathcal{R}(S)$ denote the fraction of RR sets in $\mathcal{R}$ covered by $S$. Eq. 7 guarantees a probability of at least $1 - n^l / \binom{n}{k}$ that the inequality holds:

$$|n \cdot F_\mathcal{R}(S) - \mathbb{I}_\mathbb{C}(S)| < \frac{\varepsilon}{2} \cdot OPT. \tag{8}$$

The lower bound of OPT is estimated based on the expected influence of size-$k$ random node seeds. In addition, Tang et al. [12] further minimize the number of RR sets. Nguyen et al. [13] propose SSA and D-SSA to select seeds in the node selection phase, followed by the usage of a validation method to verify the performance of the selected seeds.

## B    Parameter estimation of $OPT_{\bar{f}}$

In this part, we evaluate the parameters $\theta$ and $OPT_{\bar{f}}$. Evaluating $OPT_{\bar{f}}$ directly is computationally expensive when only a limited number of RR sets are generated. Instead, we calculate a lower bound. Initially, we randomly sample $k$ nodes as initially activated seeds, ensuring that any duplicates are removed. The probability of choosing a node $v$ is proportional to its in-degree within graph $G$. By generating a sample of $k$ nodes, we can estimate the expected spread, denoted as $KPT$, which serves as the lower bound for $OPT_{\bar{f}}$, $KPT \leq OPT_{\bar{f}}$. Let $\mathcal{R}$ denote the set of all RR sets generated by randomly selecting a super node $v$ with a probability of $\frac{w_v}{\eta}$. Let $EN$ be the expected number of coin flips required to generate a random RR set. We have the following lemma:

**Lemma 6.** $\frac{n}{m} \cdot EN = \mathbb{E}[\bar{f}(v^*)]$, where the expectation of $\bar{f}(v^*)$ is taken over the randomness of $v^*$ and the cascade process.

*Proof*: Let $R$ be a RR set by randomly selecting a super node $v$ with a probability of $\frac{w_v}{\eta}$, $p_R$ be the probability that a randomly selected edge from $G$ points to a node in $R$. Then $EN = \mathbb{E}[p_R \cdot m]$, where the expectation is taken over the random choices of $R$.

Let $v^*$ be a random node that is sampled with probability proportional to its in-degree within graph $G$, and $b(v*, R)$ be a boolean function that returns 1 if $v^* \in R$, and 0 otherwise. Then, for a fixed $R$, we have

$$p_R = \sum_{v^*} (Pr[v^*] \cdot b(v*, R)). \tag{9}$$

The probability that a seed node $v^*$ activates a randomly selected super node is

$$p_{v^*} = \sum_R (Pr[R] \cdot b(v^*, R)). \tag{10}$$

Hence, we have $\mathbb{E}[p_{v^*}] = \mathbb{E}[\bar{f}(\{v^*\})]/\eta$ and

$$
\begin{aligned}
EN/m = \mathbb{E}[p_R] &= \sum_R (Pr[R] \cdot p_R) \\
&= \sum_R (Pr[R] \cdot \sum_{v^*} (Pr[v^*] \cdot b(v^*, R))) \\
&= \sum_{v^*} (Pr[v^*] \cdot \sum_R (Pr[R] \cdot b(v^*, R))) \\
&= \sum_{v^*} (Pr[v^*] \cdot p_{v^*}) \\
&= \mathbb{E}[p_{v^*}] = \mathbb{E}[\bar{f}(\{v^*\})]/\eta,
\end{aligned}
\tag{11}
$$

which completes the proof. $\square$

Lemma 6 represents the generation complexity of RR sets, which is degenerated into the TIM solution [14] when $w_j = 1, \forall j = 1, 2, .., h$ in the classical IM problem.

We then define the width of an RR set $R_i$, denoted as $\varpi(R_i)$, to be the number of directed edges in graph $G$ that point to the nodes within $R_i$. Mathematically, this is expressed as:

$$\varpi(R) = \sum_{v \in R_i} (the \quad indegree \quad of \quad v). \tag{12}$$

**Lemma 7**. Let $R_v$ be a random RR set starting from a random super node $v$. The selection of the super node $v$ is based on the probability $\frac{w_v}{\eta}$. We define $\kappa(R)$ as follows:

$$\kappa(R) = 1 - (1 - \frac{\varpi(R)}{m})^k. \tag{13}$$

Then, $KPT = \eta \cdot \mathbb{E}_S[\mathbb{E}_{v \sim \frac{w_v}{\eta}}[\kappa(R_v)]]$, where the expectation is taken over the random choices of $R$.

*Proof*: Let $S$ be a random set of nodes sampled based on their in-degree in $G$, excluding duplicates. Let $R_v$ be a random RR set starting from super node $v$, and $\alpha_{R_v}$ be the probability of overlap between $S$ and $R_v$. The $KPT$ can be expressed as:

$$KPT = \mathbb{E}_S[\sum_{v \in T} w_v \cdot Pr[\mu(v) = 1]] = \eta \cdot \mathbb{E}_S[\mathbb{E}_{v \sim \frac{w_v}{\eta}}[\alpha_{R_v}]]. \tag{14}$$

Consider that we sample $k$ times uniformly from the edges in $G$. Let $E^*$ be the set of sampled edges without duplicates. Let $\alpha'_{R_v}$ represent the probability that one of the edges in $E^*$ points to a node in $R$. It can be observed that $\alpha'_{R_v} = \alpha_{R_v}$. Given that there are $\varpi(R_v)$ edges pointing to nodes in $R_v$, we have $\alpha'_{R_v} = 1 - (1 - \varpi(R)/m)^k = \kappa(R_v)$. Hence, the $KPT$ can be further simplified as:

$$KPT = \eta \cdot \mathbb{E}_S[\mathbb{E}_{v \sim \frac{w_v}{\eta}}[\alpha'_{R_v}]] = \eta \cdot \mathbb{E}_S[\mathbb{E}_{v \sim \frac{w_v}{\eta}}[\kappa(R_v)]], \tag{15}$$

which completes the proof. $\square$

Based on Lemma 7, we present an efficient approach for evaluating $KPT$ in Algorithm 2: We first generate $c_i$ RR sets (lines 5–6) and accumulate the corresponding $\kappa(R_v)$ (lines 7–8), and then evaluate the quality of the estimated $KPT^*$ (lines 10–12). If the estimated $KPT^*$ is less than a threshold, we repeat the process (lines 1–13). Algorithm 2 runs in at most $\log_2 \eta - 1$ iterations. Next, we analyze the time complexity of Algorithm 2.

**Theorem 5**. For $\eta \geq 2$ and $l \geq 1/2$, Algorithm 2 returns a result $KPT^* \in [KPT/4, OPT_{\bar{f}}]$ with a probability of at least $1 - \eta^{-l}$. Furthermore, the algorithm runs in the expected time complexity of $O(l(m + \eta)log\eta)$.

**Algorithm 3:** KptEstimation $(G, k)$.

---

1 **for** $i = 1 : \log_2 \eta - 1$ **do**
2      Let $c_i = (6l \log \eta + 6 \log(\log_2 \eta)) \cdot 2^i$;
3      Let $sum = 0$;
4      **for** $j = 1 : c_i$ **do**
5          Randomly choose a super node $v$ with probability $\frac{w_v}{\eta}$;
6          Generate a random RR set $R_v$.
7          $\kappa(R_v) = 1 - (1 - \frac{\varpi(R)}{m})^k$;
8          $sum = sum + \kappa(R_v)$;
9      **end**
10      **if** $sum/c_i > 1/2^i$ **then**
11          return $KPT^* = \eta \cdot \cdot sum/(2c_i)$;
12      **end**
13 **end**
14 return $KPT^* = 1$;

---

*Proof*: Reference [14] has proved that for $\eta \geq 2$ and $l \geq 1/2$, Algorithm 2 returns a result $KPT^* \in [KPT/4, OPT_{\bar{f}}]$ with a probability of at least $1 - \eta^{-l}$. The remaining problem is the time complexity, because the generation of RR sets is slightly different from the TIM method [14].

Suppose that $KPT/\eta \in [2^{-j}, 2^{-j+1}]$, the expected total number of RR sets generated by Algorithm 2 is less than $2c_{j+2} \in O(2^j l \log \eta)$ [14]. We have

$$
\begin{aligned}
O(2c_{j+2} \cdot EN) &= O(2^j l \log \eta \cdot EN) \\
&= O(2^j l \log \eta \cdot (1 + \frac{m}{\eta}) \cdot KPT) \\
&= O(2^j l \log \eta \cdot (\eta + m) \cdot 2^{-j}) \\
&= O(l(\eta + m) \log \eta),
\end{aligned}
\tag{16}
$$

which completes the proof. $\square$

## C  Other Proofs

***Proof of Theorem 1***: Consider an instance of the NP-complete problem of *Vertex Cover* in an undirected graph $G = (V, E)$. Given an integer $k$, the goal is to determine whether there exists a set $S$ of size $k$ in $V$ such that every edge in $E$ has at least one endpoint in $S$. We then show that the *Vertex Cover* problem is a specific instance of the MOIM problem.

To establish this connection, we define a motif consisting of a pair of adjacent nodes, and hence two adjacent nodes form a motif. We treat a motif is activated if all nodes in the motif are activated. If there exists a vertex cover $S$ of size $k$, it guarantees that every edge has at least one endpoint in $S$, enabling the activation of all nodes in $V$ using the linear threshold cascade model. This is the only way that corresponds to get a set $S$ with $\mathbb{I}_\mathbb{C}(S) = m$, where $m$ denotes the number of activated motifs(i.e., the number of edges). Consequently, an instance of NP-complete *Vertex Cover* problem can be viewed as a specific case of the MOIM problem, which arrives at Theorem 1. $\square$

***Proof of Lemma 5***: In Theorem 5, the function $KptEstimation$ costs expected time $O(l(m + \eta)log\eta)$. The time complexity of function $NodeSelection$ in Algorithm 3 is $O(\theta \cdot EN)$. Based on Theorem 5, we have

$$
\begin{aligned}
O(\theta \cdot EN) &= O((8 + 2\varepsilon)\eta \cdot \frac{l \log \eta + \log \binom{\eta}{k} + log2}{\varepsilon^2} \frac{EN}{OPT}) \\
&= O((8 + 2\varepsilon)\eta \cdot \frac{l \log \eta + \log \binom{\eta}{k} + log2}{\varepsilon^2} \cdot \frac{m}{\eta}) \\
&= O((k + l)(m + \eta) \log \eta / \varepsilon^2),
\end{aligned}
\tag{17}
$$

which completes the proof. $\square$

# D Generalization to other cascade models

We mainly discuss the generalization to the independent cascade (IC) model and triggering model [3], since the two models are widely used in the IM problem.

In the IC model, when a node $u$ is first activated at time $t$, for each directed edge $(u, v)$ from node $u$ to $v$, $u$ has a probability $p(u, v)$ to activate $v$ at time $t + 1$. After time $t + 1$, $u$ cannot activate any node. In the triggering model, for each node $u$, we take a sample from its in-neighbors based on some prior distribution and define the sample as the triggering set of $u$. In the cascade process, when a node $u$ is activated at time $t$, if $u$ appears in the triggering set of $v$, $v$ will be activated at time $t + 1$. Note that the IC model and LT model are two particular cases of triggering model.

In our algorithms and the analysis, we do not rely on anything specific to the LT model. The only difference is the generation of random RR sets. Reference [14] analyzed the complexity of generating RR sets of the three models. Though the three models(LT, IC, and triggering models) have slightly different processes to generate the RR sets, the time complexity of generating RR set is almost the same. Hence, our method could be generalized to the IC model and triggering model. A similar generalization in the classical IM has been shown in ref. [14], and our method follows almost the same procedure to be extended to IC and triggering models.

